# A Model of Early Visual Processing

**Laurent Itti, Jochen Braun, Dale K. Lee and Christof Koch**
{itti, achim, jjwen, koch}@klab.caltech.edu
Computation & Neural Systems, MSC 139-74
California Institute of Technology, Pasadena, CA 91125, U.S.A.

## Abstract

We propose a model for early visual processing in primates. The model consists of a population of linear spatial filters which interact through non-linear excitatory and inhibitory pooling. Statistical estimation theory is then used to derive human psychophysical thresholds from the responses of the entire population of units. The model is able to reproduce human thresholds for contrast and orientation discrimination tasks, and to predict contrast thresholds in the presence of masks of varying orientation and spatial frequency.

## 1 INTRODUCTION

A remarkably wide range of human visual thresholds for spatial patterns appears to be determined by the earliest stages of visual processing, namely, orientation- and spatial frequency-tuned visual filters and their interactions [18, 19, 3, 22, 9]. Here we consider the possibility of quantitatively relating arbitrary spatial vision thresholds to a single computational model. The success of such a unified account should reveal the extent to which human spatial vision indeed reflects one particular stage of processing. Another motivation for this work is the controversy over the neural circuits that generate orientation and spatial frequency tuning in striate cortical neurons [13, 8, 2]. We think it is likely that behaviorally defined visual filters and their interactions reveal at least some of the characteristics of the underlying neural circuitry. Two specific problems are addressed: (i) what is the minimal set of model components necessary to account for human spatial vision, (ii) is there a general decision strategy which relates model responses to behavioral thresholds and which obviates case-by-case assumptions about the decision strategy in different behavioral situations. To investigate these questions, we propose a computational model articulated around three main stages: first, a population of bandpass linear filters extracts visual features from a stimulus; second, linear filters interact through non-linear excitatory and inhibitory pooling; third, a noise model and decision strategy are assumed in order to relate the model's output to psychophysical data.

## 2  MODEL

We assume spatial visual filters tuned for a variety of orientations $\theta \in \Theta$ and spatial periods $\lambda \in \Lambda$. The filters have overlapping receptive fields in visual space. Quadrature filter pairs, $F_{\lambda,\theta}^{even}$ and $F_{\lambda,\theta}^{odd}$, are used to compute a phase-independent linear energy response, $E_{\lambda,\theta}$, to a visual stimulus $S$. A small constant background activity, $\epsilon$, is added to the linear energy responses:

$$E_{\lambda,\theta} = \sqrt{(F_{\lambda,\theta}^{even} * S)^2 + (F_{\lambda,\theta}^{odd} * S)^2} + \epsilon$$

Filters have separable Gaussian tuning curves in orientation and spatial frequency. Their corresponding shape in visual space is close to that of Gabor filters, although not separable along spatial dimensions.

### 2.1  Pooling: self excitation and divisive inhibition

A model based on linear filters alone would not correctly account for the non-linear response characteristics to stimulus contrast which have been observed psychophysically [19]. Several models have consequently introduced a non-linear transducer stage following each linear unit [19]. A more appealing possibility is to assume a non-linear pooling stage [6, 21, 3, 22]. In this study, we propose a pooling strategy inspired by Heeger's model for gain control in cat area V1 [5, 6]. The pooled response $R_{\lambda,\theta}$ of a unit tuned for $(\lambda, \theta)$ is computed from the linear energy responses of the entire population:

$$R_{\lambda,\theta} = \frac{E_{\lambda,\theta}^\gamma}{S^\delta + \sum_{\lambda',\theta'} W_{\lambda,\theta}(\lambda', \theta') E_{\lambda',\theta'}^\delta} + \eta \tag{1}$$

where the sum is taken over the entire population and $W_{\lambda,\theta}$ is a two-dimensional Gaussian weighting function centered around $(\lambda, \theta)$, and $\eta$ a background activity. The numerator in **Eq. 1** represents a non-linear self-excitation term. The denominator represents a divisive inhibitory term which depends not only on the activity of the unit $(\lambda, \theta)$ of interest, but also on the responses of other units. We shall see in **Section 3** that, in contrast to Heeger's model for electrophysiological data in which all units contribute equally to the pool, it is necessary to assume that only a subpopulation of units with tuning close to $(\lambda, \theta)$ contribute to the pool in order to account for psychophysical data. Also, we assume $\gamma > \delta$ to obtain a power law for high contrasts [7], as opposed to Heeger's physiological model in which $\gamma = \delta = 2$ to account for neuronal response saturation at high contrasts.

Several interesting properties result from this pooling model. First, a sigmoidal transducer function – in agreement with contrast discrimination psychophysics – is naturally obtained through pooling and thus need not be introduced *post-hoc*. The transducer slope for high contrasts is determined by $\gamma - \delta$, the location of its inflexion point by $S$, and the slope at this point by the absolute value of $\gamma$ (and $\delta$). Second, the tuning curves of the pooled units for orientation and spatial period do not depend of stimulus contrast, in agreement with physiological and psychophysical evidence [14]. In comparison, a model which assumes a non-linear transducer but no pooling exhibits sharper tuning curves for lower contrasts. Full contrast independence of the tuning is achieved only when all units participate in the inhibitory pool; when only sub-populations participate in the pool, some contrast dependence remains.

### 2.2  Noise model: Poisson$^\alpha$

It is necessary to assume the presence of noise in the system in order to be able to derive psychophysical performance from the responses of the population of pooled

units. The deterministic response of each unit then represents the mean of a randomly distributed "neuronal" response which varies from trial to trial in a simulated psychophysical experiment.

Existing models usually assume constant noise variance in order to simplify the subsequent decision stage [18]. Using the decision strategy presented below, it is however possible to derive psychophysical performance with a noise model whose variance increases with mean activity, in agreement with electrophysiology [16]. In what follows, Poisson$^\alpha$ noise will be assumed and approximated by a Gaussian random variable with *variance = mean$^\alpha$* ($\alpha$ is a constant close to unity).

## 2.3   Decision strategy

We use tools from statistical estimation theory to compute the system's behavioral response based on the responses of the population of pooled units. Similar tools have been used by Seung and Sompolinsky [12] under the simplifying assumption of purely Poisson noise and for the particular task of orientation discrimination in the limit of an infinite population of oriented units. Here, we extend this framework to the more general case in which any stimulus attribute may differ between the two stimulus presentations to be discriminated by the model. Let's assume that we want to estimate psychophysical performance at discriminating between two stimuli which differ by the value of a stimulus parameter $\zeta$ (e.g. contrast, orientation, spatial period).

The central assumption of our decision strategy is that the brain implements an *unbiased efficient statistic* $T(\mathcal{R};\zeta)$, which is an estimator of the parameter $\zeta$ based on the population response $\mathcal{R} = \{R_{\lambda,\theta}; \lambda \in \Lambda, \theta \in \Theta\}$. The efficient statistic is the one which, among all possible estimators of $\zeta$, has the property of minimum variance in the estimated value of $\zeta$. Although we are not suggesting any putative neuronal correlate for $T$, it is important to note that the assumption of efficient statistic does not require $T$ to be prohibitively complex; for instance, a maximum likelihood estimator proposed in the decision stage of several existing models is asymptotically (with respect to the number of observations) a efficient statistic.

Because $T$ is efficient, it achieves the Cramér-Rao bound [1]. Consequently, when the number of observations (i.e. simulated psychophysical trials) is large,

$$E[T] = \zeta \qquad \text{and} \qquad var[T] = 1/\mathcal{J}(\zeta)$$

where $E[.]$ is the mean over all observations, $var[.]$ the variance, and $\mathcal{J}(\zeta)$ is the Fisher information. The Fisher information can be computed using the noise model assumption and tuning properties of the pooled units: for a random variable $X$ with probability density $f(x;\zeta)$, it is given by [1]:

$$\mathcal{J}(\zeta) = E\left[\frac{\partial}{\partial \zeta} \ln f(x;\zeta)\right]^2$$

For our Poisson$^\alpha$ noise model and assuming that different pooled units are independent [15], this translates into:

**One unit** $R_{\lambda,\theta}$:   $J_{\lambda,\theta}(\zeta) = \left(\frac{\partial R_{\lambda,\theta}}{\partial \zeta}\right)^2 \left[\frac{1}{R_{\lambda,\theta}^\alpha} + \frac{\alpha^2}{2R_{\lambda,\theta}^2}\right]$

**All independent units:**   $\mathcal{J}(\zeta) = \sum_{\lambda,\theta} J_{\lambda,\theta}(\zeta)$

The Fisher information computed for each pooled unit and three types of stimulus parameters $\zeta$ is shown in **Figure 1**. This figure demonstrates the importance of using information from all units in the population rather than from only one unit optimally tuned for the stimulus: although the unit carrying the most information about contrast is the one optimally tuned to the stimulus pattern, more information

about orientation or spatial frequency is carried by units which are tuned to flanking orientations and spatial periods and whose tuning curves have maximum slope for the stimulus rather than maximum absolute sensitivity. In our implementation, the derivatives of pooled responses used in the expression of Fisher information are computed numerically.

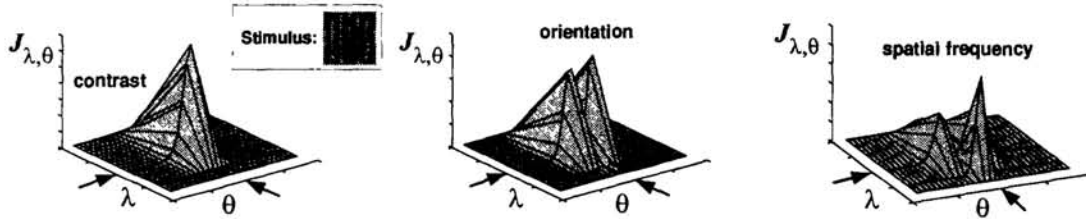

Figure 1: Fisher information computed for contrast, orientation and spatial frequency. Each node in the tridimensional meshes represents the Fisher information for the corresponding pooled unit $(\lambda, \theta)$ in a model with 30 orientations and 4 scales. Arrows indicate the unit $(\lambda, \theta)$ optimally tuned to the stimulus. The total Fisher information in the population is the sum of the information for all units.

Using the estimate of $\zeta$ and its variance from the Fisher information, it is possible to derive psychophysical performance for a discrimination task between two stimuli with parameters $\zeta_1 \le \zeta_2$ using standard ideal observer signal discrimination techniques [4]. For such discrimination, we use the Central Limit Theorem (in the limit of large number of trials) to model the noisy responses of the system as two Gaussians with means $\zeta_1$ and $\zeta_2$, and variances $\sigma_1^2 = 1/\mathcal{J}(\zeta_1)$ and $\sigma_2^2 = 1/\mathcal{J}(\zeta_2)$ respectively. A decision criterion $D$ is chosen to minimize the overall probability of error; since in our case $\sigma_1 \ne \sigma_2$ in general, we derive a slightly more complicated expression for performance $P$ at a Yes/No (one alternative forced choice) task than what is commonly used with models assuming constant noise [18]:

$$D = \frac{\zeta_2 \sigma_1^2 - \zeta_1 \sigma_2^2 - \sigma_1 \sigma_2 \sqrt{(\zeta_1 - \zeta_2)^2 + 2(\sigma_1^2 - \sigma_2^2)\log(\sigma_1/\sigma_2)}}{\sigma_1^2 - \sigma_2^2}$$

$$P = \frac{1}{2} + \frac{1}{4}\mathrm{erf}\left(\frac{\zeta_2 - D}{\sigma_2\sqrt{2}}\right) + \frac{1}{4}\mathrm{erf}\left(\frac{D - \zeta_1}{\sigma_1\sqrt{2}}\right)$$

where erf is the Normal error function. The expression for $D$ extends by continuity to $D = (\zeta_2 - \zeta_1)/2$ when $\sigma_1 = \sigma_2$. This decision strategy provides a unified, task-independent framework for the computation of psychophysical performance from the deterministic responses of the pooled units. This strategy can easily be extended to allow the model to perform discrimination tasks with respect to additional stimulus parameters, under exactly the same theoretical assumptions.

## 3  RESULTS

### 3.1  Model calibration

The parameters of the model were automatically adjusted to fit human psychophysical thresholds measured in our laboratory [17] for contrast and orientation discrimination tasks **(Figure 2)**. The model used in this experiment consisted of 60 orientations evenly distributed between 0 and 180deg. One spatial scale at 4 cycles per degree (cpd) was sufficient to account for the data. A multidimensional simplex method with simulated annealing overhead was used to determine the best fit of the model to the data [10]. The free parameters adjusted during the automatic

fits were: the noise level $\alpha$, the pooling exponents $\gamma$ and $\delta$, the inhibitory pooling constant $S$, and the background firing rates, $\epsilon$ and $\eta$.

The error function minimized by the fitting algorithm was a weighted average of three constraints: 1) least-square error with the contrast discrimination data in **Figure 2.a**; 2) least-square error with the orientation discrimination data in **Figure 2.b**; 3) because the data was sparse in the "dip-shaped" region of the curve in **Figure 2.a**, and unreliable due to the limited contrast resolution of the display used for the psychophysics, we added an additional constraint favoring a more pronounced "dip", as has been observed by several other groups [11, 19, 22].

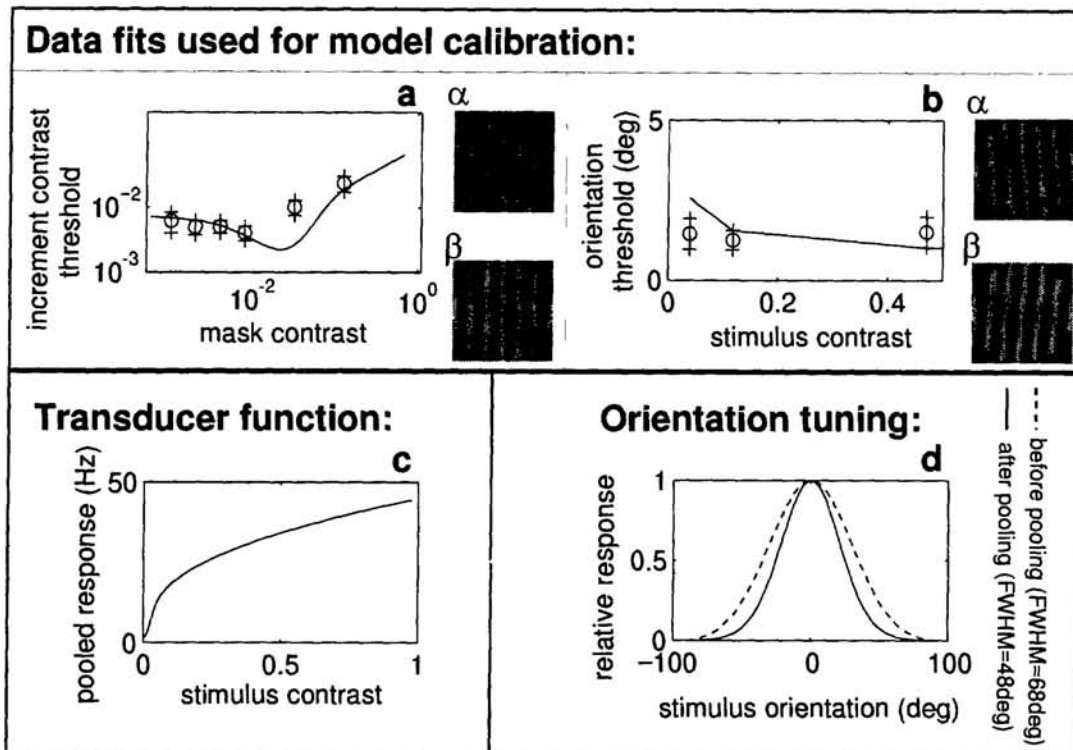

Figure 2: The model (solid lines) was calibrated using data from two psychophysical experiments: **(a)** discrimination between a pedestal contrast **(a.$\alpha$)** and the same pedestal plus an increment contrast **(a.$\beta$)**; **(b)** discrimination between two orientations near vertical **(b.$\alpha$** and **b.$\beta$)**. After calibration, the transducer function of each pooled unit **(c)** correctly exhibits an accelerating non-linearity near threshold (contrast $\approx 1\%$) and compressive non-linearity for high contrasts (Weber's law). We can see in **(d)** that pooling among units with similar tuning properties sharpens their tuning curves. Model parameters were: $\alpha \approx 0.75, \gamma \approx 4, \delta \approx 3.5, \epsilon \approx 1\%, \eta \approx 1.7Hz, S$ such that transducer inflexion point is at $4\times$ detection threshold contrast, orientation tuning FWHM=68deg (full width at half maximum), orientation pooling FWHM=40deg.

Two remaining parameters are the orientation tuning width, $\sigma_\theta$, of the filters and the width, $\sigma_{W_\theta}$, of the pool. It was not possible from the data in **Figure 2** alone to unambiguously determine these parameters. However, for any given $\sigma_\theta$, $\sigma_{W_\theta}$ is uniquely determined by the following two qualitative constraints: first, a small pool size is not desirable because it yields contrast-dependent orientation tuning; it however appears from the data in **Figure 2.b** that this tuning should not vary much over a wide range of contrasts. The second constraint is qualitatively derived from **Figure 3.a**: for large pool sizes, the model predicted significant interference between mask and test patterns even for large orientation differences. Such inter-

ference was not observed in the data for orientation differences larger than 45deg. It consequently seems that a partial inhibitory pool, composed only of a fraction of the population of oriented filters with tuning similar to the central excitatory unit, accounts best for the psychophysical data. Finally, $\sigma_\theta$ was fixed so as to yield a correct qualitative curve shape for **Figure 3.a**.

## 3.2 Predictions

We used complex stimuli from masking experiments to test the predictive value of the model (**Figure 3**). Although it was necessary to use some of the qualitative properties of the data seen in **Figure 3.a** to calibrate the model as detailed above, the calibrated model correctly produced a quantitative fit of this data. The calibrated model also correctly predicted the complex data of **Figure 3.b**.

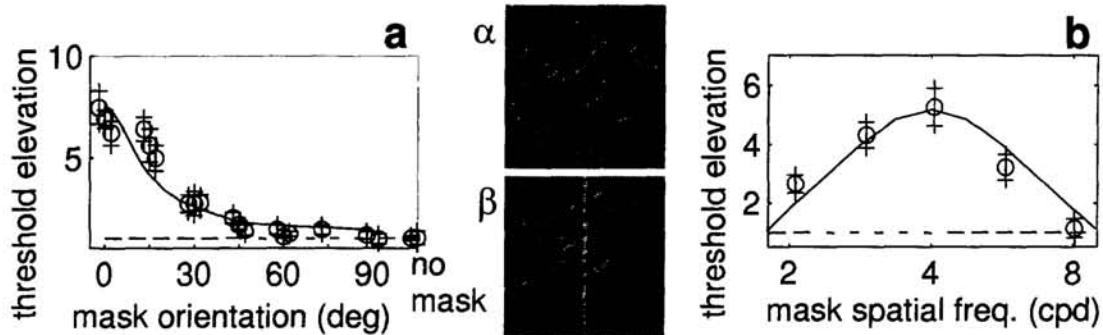

Figure 3: Prediction of psychophysical contrast thresholds in the presence of an oblique mask. The mask was a 50%-contrast stochastic oriented pattern ($\alpha$), and the superimposed test pattern was a sixth-derivative of Gaussian bar ($\beta$). In (**a**), threshold elevation (i.e. ratio of threshold in the presence of mask to threshold in the absence of mask) was measured for varying mask orientation, for mask and test patterns at 4 cycles per degree (cpd). In (**b**), orientation difference between test and mask was fixed to 15deg, and threshold elevation was measured as a function of mask spatial frequency. Solid lines represent model predictions, and dashed lines represent unity threshold elevation.

## 4 DISCUSSION AND CONCLUSION

We have developed a model of early visual processing in humans which accounts for a wide range of measured spatial vision thresholds and which predicts behavioral thresholds for a potentially unlimited number of spatial discriminations. In addition to orientation- and spatial-frequency-tuned units, we have found it necessary to assume two types of interactions between such units: (i) non-linear self-excitation of each unit and (ii) divisive normalization of each unit response relative to the responses of similarly tuned units. All model parameters are constrained by psychophysical data and an automatic fitting procedure consistently converged to the same parameter set regardless of the initial position in parameter space.

Our two main contributions are the small number of model components and the unified, task-independent decision strategy. Rather than making different assumptions about the decision strategy in different behavioral tasks, we combine the information contained in the responses of all model units in a manner that is optimal for any behavioral task. We suggest that human observers adopt a similarly optimal decision procedure as they become familiar with a particular task ("task set"). Although here we apply this decision strategy only to the discrimination of stimulus contrast, orientation, and spatial frequency, it can readily be generalized to arbitrary discriminations such as, for example, the discrimination of vernier targets.

So far we have considered only situations in which the same decision strategy is optimal for every stimulus presentation. We are now studying situations in which the optimal decision strategy varies unpredictably from trial to trial ("decision uncertainty"). For example, situations in which the observer attempts to detect an increase in either the spatial frequency or the contrast of stimulus. In this way, we hope to learn the extent to which our model reflects the decision strategy adopted by human observers in an even wider range of situations. We have also assumed that the model's units were independent, which is not strictly true in biological systems (although the main source of correlation between neurons is the overlap between their respective tuning curves, which is accounted for in the model). The mathematical developments necessary to account for fixed or variable covariance between units are currently under study.

In contrast to other models of early visual processing [5, 6], we find that the psychophysical data is consistent only with interactions between similarly tuned units (e.g., "near-orientation inhibition"), not with interactions between units of very different tuning (e.g., "cross-orientation inhibition"). Although such partial pooling does not render tuning functions completely contrast-independent, an additional degree of contrast-independence could be provided by pooling across different spatial locations. This issue is currently under investigation.

In conclusion, we have developed a model based on self-excitation of each unit, divisive normalization [5, 6] between similarly tuned units, and an ideal observer decision strategy. It was able to reproduce a wide range of human visual thresholds. The fact that such a simple and idealized model can account quantitatively for a wide range of psychophysical observations greatly strengthens the notion that spatial vision thresholds reflect processing at one particular neuroanatomical level.

**Acknowledgments:** This work was supported by NSF-Engineering Research Center (ERC), NIMH, ONR, and the Sloan Center for Theoretical Neurobiology.

# References

[1]  Cover TM, Thomas JA. Elem Info Theo, Wiley & Sons, 1991
[2]  Ferster D, Chung S, Wheat H. *Nature* 1996;380(6571):249-52
[3]  Foley JM. *J Opt Soc A* 1994;11(6):1710-9
[4]  Green DM, Swets JA. Signal Detectability and Psychophys. Wiley & Sons, 1966.
[5]  Heeger DJ. Comput Models of Vis Processing, MIT Press, 1991
[6]  Heeger DJ. *Vis Neurosci* 1992;9:181-97
[7]  Nachmias J, Sansbury RV. *Vis Res* 1974;14:1039-42
[8]  Nelson S, Toth L, Sheth B, Sur M. *Science* 1994;265(5173):774-77
[9]  Perona P, Malik J. *J Opt Soc A* 1990;7(5):923-32
[10] Press WH, Teukolsky SA, *et al.* Num Rec in C. Cambridge University Press, 1992
[11] Ross J, Speed HD. *Proc R Soc B* 1991;246:61-9
[12] Seung HS, Sompolinksy H. *Proc Natl Acad Sci USA* 1993;90:10749-53.
[13] Sillito AM. *Progr Brain Res* 1992;90:349-84
[14] Skottun BC, Bradley A, Sclar G et al. *J Neurophys* 1987;57(3):773-86
[15] Snippe HP, Koenderink JJ. *Biol Cybern* 1992;67:183-90
[16] Teich MC, Turcott RG, Siegel RM. *IEEE Eng Med Biol* 1996;Sept-Oct,79-87
[17] Wen J, Koch C, Braun J. *Proc ARVO* 1997;5457
[18] Wilson HR, Bergen JR. *Vis Res* 1979;19:19-32
[19] Wilson HR. *Biol Cybern* 1980;38:171-8
[20] Wilson HR, McFarlane DK, Phillips GC. *Vis Res* 1983;23;873-82.
[21] Wilson HR, Humanski R. *Vis Res* 1993;33(8):1133-50
[22] Zenger B, Sagi D. *Vis Res* 1996;36(16):2497-2513.
